# Does the Wake-sleep Algorithm Produce Good Density Estimators?

**Brendan J. Frey, Geoffrey E. Hinton**
Department of Computer Science
University of Toronto
Toronto, ON M5S 1A4, Canada
{frey, hinton}@cs.toronto.edu

**Peter Dayan**
Department of Brain and Cognitive Sciences
Massachusetts Institute of Technology
Cambridge, MA 02139, USA
dayan@ai.mit.edu

## Abstract

The wake-sleep algorithm (Hinton, Dayan, Frey and Neal 1995) is a relatively efficient method of fitting a multilayer stochastic generative model to high-dimensional data. In addition to the top-down connections in the generative model, it makes use of bottom-up connections for approximating the probability distribution over the hidden units given the data, and it trains these bottom-up connections using a simple delta rule. We use a variety of synthetic and real data sets to compare the performance of the wake-sleep algorithm with Monte Carlo and mean field methods for fitting the same generative model and also compare it with other models that are less powerful but easier to fit.

## 1 INTRODUCTION

Neural networks are often used as bottom-up recognition devices that transform input vectors into representations of those vectors in one or more hidden layers. But multilayer networks of stochastic neurons can also be used as top-down generative models that produce patterns with complicated correlational structure in the bottom visible layer. In this paper we consider generative models composed of layers of stochastic binary logistic units. Given a generative model parameterized by top-down weights, there is an obvious way to perform unsupervised learning. The generative weights are adjusted to maximize the probability that the visible vectors generated by the model would match the observed data. Unfortunately, to compute the derivatives of the log probability of a visible vector, $d$, with respect to the generative weights, $\theta$, it is necessary to consider all possible ways in which $d$ could be generated. For each possible binary representation $\alpha$ in the hidden units the derivative needs to be weighted by the posterior probability of $\alpha$ given $d$ and $\theta$:

$$P(\alpha|d, \theta) = P(\alpha|\theta)P(d|\alpha, \theta)/\sum_{\beta} P(\beta|\theta)P(d|\beta, \theta). \tag{1}$$

It is intractable to compute $P(\alpha|d, \theta)$, so instead of minimizing $-\log P(d|\theta)$, we minimize an easily computed upper bound on this quantity that depends on some additional parameters, $\phi$:

$$-\log P(d|\theta) \leq F(d|\theta, \phi) = -\sum_\alpha Q(\alpha|d, \phi)\log P(\alpha, d|\theta) + \sum_\alpha Q(\alpha|d, \phi)\log Q(\alpha|d, \phi). \quad (2)$$

$F(d|\theta, \phi)$ is a Helmholtz free energy and is equal to $-\log P(d|\theta)$ when the distribution $Q(\bullet|d, \phi)$ is the same as the posterior distribution $P(\bullet|d, \theta)$. Otherwise, $F(d|\theta, \phi)$ exceeds $-\log P(d|\theta)$ by the asymmetric divergence:

$$D = \sum_\alpha Q(\alpha|d, \phi)\log(Q(\alpha|d, \phi)/P(\alpha|d, \theta)). \quad (3)$$

We restrict $Q(\bullet|d, \phi)$ to be a product distribution within each layer that is conditional on the binary states in the layer below and we can therefore compute it efficiently using a bottom-up recognition network. We call a model that uses bottom-up connections to minimize the bound in equation 2 in this way a *Helmholtz machine* (Dayan, Hinton, Neal and Zemel 1995). The recognition weights $\phi$ take the binary activities in one layer and stochastically produce binary activities in the layer above using a logistic function. So, for a given visible vector, the recognition weights may produce many different representations in the hidden layers, but we can get an unbiased sample from the distribution $Q(\bullet|d, \phi)$ in a single bottom-up pass through the recognition network.

The highly restricted form of $Q(\bullet|d, \phi)$ means that even if we use the optimal recognition weights, the gap between $F(d|\theta, \phi)$ and $-\log P(d|\theta)$ is large for some generative models. However, when $F(d|\theta, \phi)$ is minimized with respect to the generative weights, these models will generally be avoided.

$F(d|\theta, \phi)$ can be viewed as the expected number of bits required to communicate a visible vector to a receiver. First we use the recognition model to get a sample from the distribution $Q(\bullet|d, \phi)$. Then, starting at the top layer, we communicate the activities in each layer using the top-down expectations generated from the already communicated activities in the layer above. It can be shown that the number of bits required for communicating the state of each binary unit is $s_k \log(q_k/p_k) + (1-s_k)\log[(1-q_k)/(1-p_k)]$, where $p_k$ is the top-down probability that $s_k$ is on and $q_k$ is the bottom-up probability that $s_k$ is on.

There is a very simple on-line algorithm that minimizes $F(d|\theta, \phi)$ with respect to the generative weights. We simply use the recognition network to generate a sample from the distribution $Q(\bullet|d, \phi)$ and then we increment each top-down weight $\theta_{kj}$ by $\varepsilon s_k(s_j - p_j)$, where $\theta_{kj}$ connects unit $k$ to unit $j$. It is much more difficult to exactly follow the gradient of $F(d|\theta, \phi)$ with respect to the recognition weights, but there is a simple approximate method (Hinton, Dayan, Frey and Neal 1995). We generate a stochastic sample from the generative model and then we increment each bottom-up weight $\phi_{ij}$ by $\varepsilon s_i(s_j - q_j)$ to increase the log probability that the recognition weights would produce the correct activities in the layer above. This way of fitting a Helmholtz machine is called the "wake-sleep" algorithm and the purpose of this paper is to assess how effective it is at performing high-dimensional density estimation on a variety of synthetically constructed data sets and two real-world ones. We compare it with other methods of fitting the same type of generative model and also with simpler models for which there are efficient fitting algorithms.

## 2 COMPETITORS

We compare the wake-sleep algorithm with six other density estimation methods. All data units are binary and can take on values $d_k = 1$ (on) and $d_k = 0$ (off).

**Gzip.** Gzip (Gailly, 1993) is a practical compression method based on Lempel-Ziv coding. This sequential data compression technique encodes future segments of data by transmit-

ting codewords that consist of a pointer into a buffer of recent past output together with the length of the segment being coded. Gzip's performance is measured by subtracting the length of the compressed training set from the length of the compressed training set plus a subset of the test set. Taking all disjoint test subsets into account gives an overall test set code cost. Since we are interested in estimating the expected performance on *one* test case, to get a tight lower bound on gzip's performance, the subset size should be kept as small as possible in order to prevent gzip from using early test data to compress later test data.

**Base Rate Model.** Each visible unit $k$ is assumed to be independent of the others with a probability $p_k$ of being on. The probability of vector $d$ is $p(d) = \prod_k p_k^{d_k} (1 - p_k)^{1-d_k}$. The arithmetic mean of unit $k$'s activity is used to estimate $p_k$, except in order to avoid serious overfitting, one extra on and one extra off case are included in the estimate.

**Binary Mixture Model.** This method is a hierarchical extension of the base rate model which uses more than one set of base rates. Each set is called a component. Component $j$ has probability $\pi_j$ and awards each visible unit $k$ a probability $p_{jk}$ of being on. The net probability of $d$ is $p(d) = \sum_j \pi_j \prod_k p_{jk}^{d_k} (1 - p_{jk})^{1-d_k}$. For a given training datum, we consider the component identity to be a missing value which must be filled in before the parameters can be adjusted. To accomplish this, we use the expectation maximization algorithm (Dempster, Laird and Rubin 1977) to maximize the log-likelihood of the training set, using the same method as above to avoid serious overfitting.

**Gibbs Machine (GM).** This machine uses the same generative model as the Helmholtz machine, but employs a Monte Carlo method called Gibbs sampling to find the posterior in equation 1 (Neal, 1992). Unlike the Helmholtz machine it does not require a separate recognition model and with sufficiently prolonged sampling it inverts the generative model perfectly. Each hidden unit is sampled in fixed order from a probability distribution conditional on the states of the other hidden and visible units. To reduce the time required to approach equilibrium, the network is annealed during sampling.

**Mean Field Method (MF).** Instead of using a separate recognition model to approximate the posterior in equation 1, we can assume that the distribution over hidden units is factorial for a given visible vector. Obtaining a good approximation to the posterior is then a matter of minimizing free energy with respect to the mean activities. In our experiments, we use the on-line mean field learning algorithm due to Saul, Jaakkola, and Jordan (1996).

**Fully Visible Belief Network (FVBN).** This method is a special case of the Helmholtz machine where the top-down network is fully connected and there are no hidden units. No recognition model is needed since there is no posterior to be approximated.

## 3 DATA SETS

The performances of these methods were compared on five synthetic data sets and two real ones. The synthetic data sets had matched complexities: the generative models that produced them had 100 visible units and between 1000 and 2500 parameters. A data set with 100,000 examples was generated from each model and then partitioned into 10,000 for training, 10,000 for validation and 80,000 for testing. For tractable cases, each data set entropy was approximated by the negative log-likelihood of the training set under its generative model. These entropies are approximate lower bounds on the performance.

The first synthetic data set was generated by a mixture model with 20 components. Each component is a vector of 100 base rates for the 100 visible units. To make the data more realistic, we arranged for there to be many different components whose base rates are all extreme (near 0 or 1) — representing well-defined clusters — and a few components with most base rates near 0.5 — representing much broader clusters. For component $j$, we selected base rate $p_{jk}$ from a beta distribution with mean $\mu_j$ and variance $\mu_j(1-\mu_j)/40$ (we chose this variance to keep the entropy of visible units low for $\mu_j$ near 0 or 1, representing well-defined clusters). Then, as often as not we randomly replaced each $p_{jk}$ with $1-p_{jk}$ to

make each component different (without doing this, all components would favor all units off). In order to obtain many well-defined clusters, the component means $\mu_j$ were themselves sampled from a beta distribution with mean 0.1 and variance 0.02.

The next two synthetic data sets were produced using sigmoidal belief networks (Neal 1992) which are just the generative parts of binary stochastic Helmholtz machines. These networks had full connectivity between layers, one with a 20⇒100 architecture and one with a 5⇒10⇒15⇒20⇒100 architecture. The biases were set to 0 and the weights were sampled uniformly from [-2,2), a range chosen to keep the networks from being deterministic.

The final two synthetic data sets were produced using Markov random fields. These networks had full bidirectional connections between layers. One had a 10⇔20⇔100 architecture, and the other was a concatenation of ten independent 10⇔10 fields. The biases were set to 0 and the weights were sampled from the set {-4, 0, 4} with probabilities {0.4, 0.4, 0.2}. To find data sets with high-order structure, versions of these networks were sampled until data sets were found for which the base rate method performed badly.

We also compiled two versions of a data set to which the wake-sleep algorithm has previously been applied (Hinton *et al.* 1995). These data consist of normalized and quantized 8x8 binary images of handwritten digits made available by the US Postal Service Office of Advanced Technology. The first version consists of a total of 13,000 images partitioned as 6000 for training, 2000 for validation and 5000 for testing. The second version consists of *pairs* of 8x8 images (*ie.* 128 visible units) made by concatenating vectors from each of the above data sets with those from a random reordering of the respective data set.

## 4 TRAINING DETAILS

The exact log-likelihoods for the base rate and mixture models can be computed, because these methods have no or few hidden variables. For the other methods, computing the exact log-likelihood is usually intractable. However, these methods provide an approximate upper bound on the negative log-likelihood in the form of a coding cost or Helmholtz free energy, and results are therefore presented as coding costs in bits.

Because gzip performed poorly on the synthetic tasks, we did not break up the test and validation sets into subsets. On the digit tasks, we broke the validation and test sets up to make subsets of 100 visible vectors. Since the "-9" gzip option did not improve performance significantly, we used the default configuration.

To obtain fair results, we tried to automate the model selection process subject to the constraint of obtaining results in a reasonable amount of time. For the mixture model, the Gibbs machine, the mean field method, and the Helmholtz machine, a single learning run was performed with each of four different architectures using performance on a validation set to avoid wasted effort. Performance on the validation set was computed every five epochs, and if two successive validation performances were not better than the previous one by more than 0.2%, learning was terminated. The network corresponding to the best validation performance was selected for test set analysis. Although it would be desirable to explore a wide range of architectures, it would be computationally ruinous. The architectures used are given in tables 3 and 4 in the appendix.

The Gibbs machine was annealed from an initial temperature of 5.0. Between each sweep of the network, during which each hidden unit was sampled once, the temperature was multiplied by 0.9227 so that after 20 sweeps the temperature was 1.0. Then, the generative weights were updated using the delta rule. To bound the datum probability, the network is annealed as above and then 40 sweeps at unity temperature are performed while summing the probability over one-nearest-neighbor configurations, checking for overlap.

A learning rate of 0.01 was used for the Gibbs machine, the mean field method, the Helmholtz machine, and the fully visible belief network. For each of these methods, this value was found to be roughly the largest possible learning rate that safely avoided oscillations.

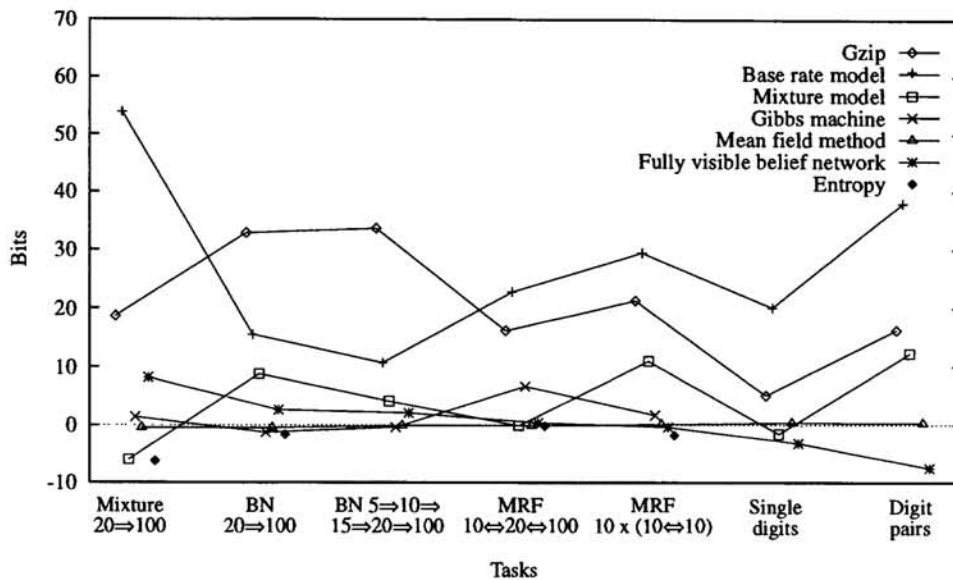

Figure 1. Compression performance relative to the Helmholtz machine. Lines connecting the data points are for visualization only, since there is no meaningful interpolant.

## 5 RESULTS

The learning times and the validation performances are given in tables 3 and 4 of the appendix. Test set appraisals and total learning times are given in table 1 for the synthetic tasks and in table 2 for the digit tasks. Because there were relatively many training cases in each simulation, the validation procedure serves to provide timing information more than to prevent overfitting. Gzip and the base rate model were very fast, followed by the fully visible belief network, the mixture model, the Helmholtz machine, the mean field method, and finally the Gibbs machine. Test set appraisals are summarized by compression performance relative to the Helmholtz machine in figure 1 above. Greater compression sizes correspond to lower test set likelihoods and imply worse density estimation. When available, the data set entropies indicate how close to optimum each method comes.

The Helmholtz machine yields a much lower cost compared to gzip and base rates on all tasks. Compared to the mixture model, it gives a lower cost on both BN tasks and the MRF 10 x (10⇔10) task. The latter case shows that the Helmholtz machine was able to take advantage of the independence of the ten concatenated input segments, whereas the mixture method was not. Simply to *represent* a problem where there are only two distinct clusters present in each of the ten segments, the mixture model would require $2^{10}$ components. Results on the two BN tasks indicate the Helmholtz machine is better able to model multiple simultaneous causes than the mixture method, which requires that only one component (cause) is active at a time. On the other hand, compared to the mixture model, the Helmholtz machine performs poorly on the Mixture 20⇒100 task. It is not able to learn that only one cause should be active at a time. This problem can be avoided by hard-wiring softmax groups into the Helmholtz machine. On the five synthetic tasks, the Helmholtz machine performs about the same as or better than the Gibbs machine, and runs two orders of magnitude faster. (The Gibbs machine was too slow to run on the digit tasks.) While the quality of density estimation produced by the mean field method is indistinguishable from the Helmholtz machine, the latter runs an order of magnitude faster than the mean field algorithm we used. The fully visible belief network performs significantly better than the Helmholtz machine on the two digit tasks and significantly worse on two of the synthetic tasks. It is trained roughly two orders of magnitude faster than the Helmholtz machine.

Table 1. Test set cost (bits) and total training time (*hrs*) for the synthetic tasks.

| | Model used to produce synthetic data | | | | | | | | | |
|---|---|---|---|---|---|---|---|---|---|---|
| | Mixture 20⇒100 | | BN 20⇒100 | | BN 5⇒10⇒ 15⇒20⇒100 | | MRF 10⇔20⇔100 | | MRF 10 x (10⇔10) | |
| Entropy | 36.5 | | 63.5 | | unknown | | 19.2 | | 36.8 | |
| gzip | 61.4 | *0* | 98.0 | *0* | 92.1 | *0* | 35.6 | *0* | 59.9 | *0* |
| Base rates | 96.6 | *0* | 80.7 | *0* | 69.2 | *0* | 42.2 | *0* | 68.1 | *0* |
| Mixture | 36.7 | *0* | 74.0 | *0* | 62.6 | *1* | 19.3 | *1* | 49.6 | *1* |
| GM | 44.1 | *131* | 63.9 | *240* | 58.1 | *251* | 26.1 | *195* | 40.3 | *145* |
| MF | 42.2 | *68* | 64.7 | *80* | 58.4 | *68* | 19.3 | *75* | 38.7 | *89* |
| HM | 42.7 | *8* | 65.2 | *3* | 58.5 | *4* | 19.4 | *2* | 38.6 | *4* |
| FVBN | 50.9 | *0* | 67.8 | *0* | 60.6 | *0* | 19.8 | *0* | 38.2 | *0* |

Table 2. Test set cost (bits) and training time (*hrs*) for the digit tasks.

| Method | Single digits | | Method | Digit pairs | |
|---|---|---|---|---|---|
| gzip | 44.3 | *0* | gzip | 89.2 | *0* |
| Base rates | 59.2 | *0* | Base rates | 118.4 | *0* |
| Mixture | 37.5 | *0* | Mixture | 92.7 | *1* |
| MF | 39.5 | *38* | MF | 80.7 | *104* |
| HM | 39.1 | *2* | HM | 80.4 | *7* |
| FVBN | 35.9 | *0* | FVBN | 72.9 | *0* |

## 6 CONCLUSIONS

If we were given a new data set and asked to leave our research biases aside and do efficient density estimation, how would we proceed? Evidently it would not be worth trying gzip and the base rate model. We'd first try the fully visible belief network and the mixture model, since these are fast and sometimes give good estimates. Hoping to extract extra higher-order structure, we would then proceed to use the Helmholtz machine or the mean field method (keeping in mind that our implementation of the Helmholtz machine is considerably faster than Saul *et al.*'s implementation of the mean field method). Because it is so slow, we would avoid using the Gibbs machine unless the data set was very small.

### Acknowledgments

We greatly appreciate the mean field software provided by Tommi Jaakkola and Lawrence Saul. We thank members of the Neural Network Research Group at the University of Toronto for helpful advice. The financial support from ITRC, IRIS, and NSERC is appreciated.

### References

Dayan, P., Hinton, G. E., Neal, R. M., and Zemel, R. S. 1995. The Helmholtz machine. *Neural Computation* 7, 889-904.

Dempster, A. P., Laird, N. M. and Rubin, D. B. 1977. Maximum likelihood from incomplete data via the EM algorithm. *J. Royal Statistical Society, Series B* 34, 1-38.

Gailly, J. 1993. gzip program for unix.

Hinton, G. E., Dayan, P., Frey, B. J., Neal, R. M. 1995. The wake-sleep algorithm for unsupervised neural networks. *Science* 268, 1158-1161.

Neal, R. M. 1992. Connectionist learning of belief networks. *Artificial Intelligence* 56, 71-113.

Saul, L. K., Jaakkola, T., and Jordan, M. I. 1996. Mean field theory for sigmoid belief networks. Submitted to *Journal of Artificial Intelligence*.

## Appendix

The average validation set cost per example and the associated learning time for each simulation are listed in tables 3 and 4. Architectures judged to be optimal according to validation performance are indicated by "*" and were used to produce the test results given in the body of this paper.

Table 3. Validation set cost (bits) and learning time (*min*) for the synthetic tasks.

| | Model used to produce synthetic data | | | | | | | | | |
|---|---|---|---|---|---|---|---|---|---|---|
| | Mixture 20⇒100 | | BN 20⇒100 | | BN 5⇒10⇒ 15⇒20⇒100 | | MRF 10⇔20⇔100 | | MRF 10 x (10⇔10) | |
| gzip | 61.6 | 0 | 98.1 | 0 | 92.3 | 0 | 35.6 | 0 | 60.0 | 0 |
| Base rates | 96.7 | 0 | 80.7 | 0 | 69.4 | 0 | 42.1 | 0 | 68.1 | 0 |
| Mixture 20⇒100 | 44.6 | 3 | 75.6 | 3 | 63.9 | 4 | 19.2* | 3 | 54.8 | 5 |
| Mixture 40⇒100 | 36.8* | 5 | 74.8 | 5 | 63.2 | 7 | 19.2 | 7 | 52.4 | 15 |
| Mixture 60⇒100 | 36.8 | 7 | 74.4 | 7 | 62.9 | 8 | 19.2 | 8 | 51.0 | 17 |
| Mixture 100⇒100 | 37.0 | 14 | 74.0* | 12 | 62.7* | 13 | 19.3 | 12 | 49.6* | 22 |
| GM 20⇒100 | 50.6 | 1187 | 63.9* | 1639 | 58.1* | 2084 | 26.1* | 934 | 40.3* | 1425 |
| GM 50⇒100 | 68.8 | 2328 | 80.4 | 3481 | 76.4 | 5234 | 49.2 | 6472 | 56.5 | 3472 |
| GM 10⇒20⇒100 | 44.1* | 872 | 66.4 | 1771 | 59.8 | 3084 | 28.0 | 767 | 42.3 | 1033 |
| GM 20⇒50⇒100 | 52.7 | 3476 | 91.3 | 7504 | 88.0 | 4647 | 55.3 | 3529 | 63.5 | 2781 |
| MF 20⇒100 | 49.5 | 518 | 64.6 | 427 | 58.4* | 497 | 19.4 | 862 | 39.2 | 471 |
| MF 50⇒100 | 49.9 | 1644 | 64.8 | 1945 | 58.6 | 1465 | 20.4 | 1264 | 38.7* | 2427 |
| MF 10⇒20⇒100 | 46.0 | 306 | 64.6* | 658 | 58.5 | 543 | 19.3* | 569 | 38.9 | 882 |
| MF 20⇒50⇒100 | 42.1* | 1623 | 65.0 | 1798 | 58.6 | 1553 | 19.3 | 1778 | 38.8 | 1575 |
| HM 20⇒100 | 50.0 | 41 | 65.2 | 28 | 58.8 | 41 | 19.7 | 15 | 38.6* | 30 |
| HM 50⇒100 | 50.7 | 81 | 65.5 | 66 | 59.4 | 78 | 20.2 | 27 | 38.9 | 46 |
| HM 10⇒20⇒100 | 43.4 | 32 | 65.1* | 38 | 58.5* | 45 | 19.4* | 21 | 38.9 | 46 |
| HM 20⇒50⇒100 | 42.6* | 308 | 67.2 | 69 | 59.2 | 93 | 19.5 | 64 | 39.4 | 102 |
| FVBN | 51.0 | 7 | 67.8 | 7 | 60.7 | 6 | 19.8 | 8 | 38.3 | 6 |

Table 4. Validation set cost (bits) and learning time (*min*) for the digit tasks.

| Method | Single digits | | Method | Digit pairs | |
|---|---|---|---|---|---|
| gzip | 44.2 | 0 | gzip | 88.8 | 1 |
| Base rates | 59.0 | 0 | Base rates | 117.9 | 0 |
| Mixture 16⇒64 | 43.2 | 1 | Mixture 32⇒128 | 96.9 | 6 |
| Mixture 32⇒64 | 40.0 | 4 | Mixture 64⇒128 | 93.8 | 8 |
| Mixture 64⇒64 | 38.0 | 5 | Mixture 128⇒128 | 92.4* | 14 |
| Mixture 128⇒64 | 37.1* | 6 | Mixture 256⇒128 | 92.8 | 27 |
| MF 16⇒24⇒64 | 39.9 | 341 | MF 16⇒24⇒32⇒128 | 82.7 | 1335 |
| MF 24⇒32⇒64 | 39.1* | 845 | MF 16⇒32⇒64⇒128 | 81.2 | 1441 |
| MF 12⇒16⇒24⇒64 | 39.8 | 475 | MF 12⇒16⇒24⇒32⇒128 | 82.8 | 896 |
| MF 16⇒24⇒32⇒64 | 39.1 | 603 | MF 12⇒16⇒32⇒64⇒128 | 80.1* | 2586 |
| HM 16⇒24⇒64 | 39.7 | 24 | HM 16⇒24⇒32⇒128 | 83.8 | 76 |
| HM 24⇒32⇒64 | 39.4 | 34 | HM 16⇒32⇒64⇒128 | 80.1* | 138 |
| HM 12⇒16⇒24⇒64 | 40.4 | 16 | HM 12⇒16⇒24⇒32⇒128 | 84.6 | 74 |
| HM 16⇒24⇒32⇒64 | 38.9* | 52 | HM 12⇒16⇒32⇒64⇒128 | 80.1 | 135 |
| FVBN | 35.8 | 1 | FVBN | 72.5 | 7 |
